# K-Local Hyperplane and Convex Distance Nearest Neighbor Algorithms

**Pascal Vincent and Yoshua Bengio**
Dept. IRO, Université de Montréal
C.P. 6128, Montreal, Qc, H3C 3J7, Canada
{*vincentp,bengioy*}*@iro.umontreal.ca*
http://www.iro.umontreal.ca/~vincentp

## Abstract

Guided by an initial idea of building a complex (non linear) decision surface with maximal *local margin* in input space, we give a possible geometrical intuition as to why K-Nearest Neighbor (KNN) algorithms often perform more poorly than SVMs on classification tasks. We then propose modified K-Nearest Neighbor algorithms to overcome the perceived problem. The approach is similar in spirit to *Tangent Distance*, but with invariances inferred from the local neighborhood rather than prior knowledge. Experimental results on real world classification tasks suggest that the modified KNN algorithms often give a dramatic improvement over standard KNN and perform as well or better than SVMs.

## 1 Motivation

The notion of *margin* for classification tasks has been largely popularized by the success of the Support Vector Machine (SVM) [2, 15] approach. The *margin* of SVMs has a nice geometric interpretation[1]: it can be defined informally as (twice) the smallest Euclidean distance between the decision surface and the closest training point. The decision surface produced by the original SVM algorithm is the hyperplane that maximizes this distance while still correctly separating the two classes. While the notion of keeping the largest possible safety *margin* between the decision surface and the data points seems very reasonable and intuitively appealing, questions arise when extending the approach to building more complex, non-linear decision surfaces.

Non-linear SVMs usually use the "kernel trick" to achieve their non-linearity. This conceptually corresponds to first mapping the input into a higher-dimensional feature space with some non-linear transformation and building a maximum-margin hyperplane (a linear decision surface) there. The "trick" is that this mapping is never computed directly, but implicitly induced by a kernel. In this setting, the margin being maximized is still the smallest Euclidean distance between the decision surface and the training points, but this time measured in some strange, sometimes infinite dimensional, kernel-induced feature space rather than the original input space. It is less clear whether maximizing the margin in this new space, is meaningful in general (see [16]).

A different approach is to try and build a non-linear decision surface with maximal distance to the closest data point as measured directly in input space (as proposed in [14]). We could for instance restrict ourselves to a certain class of decision functions and try to find the function with maximal margin among this class. But let us take this even further. Extending the idea of building a correctly separating non-linear decision surface as far away as possible from the data points, we define the notion of *local margin* as the Euclidean distance, in input space, between a given point on the decision surface and the closest training point. Now would it be possible to find an algorithm that could produce a decision surface which correctly separates the classes and such that the *local margin* is everywhere maximal along its surface? Surprisingly, the plain old Nearest Neighbor algorithm (1NN) [5] does precisely this!

So why does 1NN in practice often perform worse than SVMs? One typical explanation, is that it has too much capacity, compared to SVM, that the class of function it can produce is too rich. But, considering it has *infinite* capacity (VC-dimension), 1NN is still performing quite well... This study is an attempt to better understand what is happening, based on geometrical intuition, and to derive an improved Nearest Neighbor algorithm from this understanding.

## 2 Fixing a broken Nearest Neighbor algorithm

### 2.1 Setting and definitions

The setting is that of a classical classification problem in $\mathbb{R}^n$ (the *input space*).

We are given a *training set* $\mathcal{S}$ of $l$ points $\{x_1, \ldots, x_l\}$, $x_i \in \mathbb{R}^n$ and their corresponding class label $\{y_1 = y(x_1), \ldots, y_l = y(x_l)\}$, $y_i \in \mathcal{C}$, $\mathcal{C} = \{1, \ldots, N_c\}$ where $N_c$ is the number of different classes. The $(x, y)$ pairs are assumed to be samples drawn from an unknown distribution $P(X, Y)$. Barring duplicate inputs, the class labels associated to each $x \in \mathcal{S}$ define a partition of $\mathcal{S}$: let $\mathcal{S}_c = \{x \in \mathcal{S} \mid y(x) = c\}$.

The problem is to find a *decision function* $\tilde{f} : \mathbb{R}^n \to \mathcal{C}$ that will generalize well on new points drawn from $P(X, Y)$. $\tilde{f}$ should ideally minimize the *expected classification error*, i.e. minimize $E_P[I_{\tilde{f}(X) \neq Y}]$ where $E_P$ denotes the expectation with respect to $P(X, Y)$ and $I_{\tilde{f}(x) \neq y}$ denotes the indicator function, whose value is 1 if $\tilde{f}(x) \neq y$ and 0 otherwise.

In the previous and following discussion, we often refer to the concept of *decision surface*, also known as *decision boundary*. The function $\tilde{f}$ corresponding to a given algorithm defines for any class $c \in \mathcal{C}$ two regions of the input space: the region $R_c = \{x \in \mathbb{R}^n \mid \tilde{f}(x) = c\}$ and its complement $\mathbb{R}^n - R_c$. The *decision surface* for class $c$ is the "boundary" between those two regions, i.e. the contour of $R_c$, and can be seen as a $n-1$ dimensional manifold (a "surface" in $\mathbb{R}^n$) possibly made of several disconnected components. For simplicity, when we mention *the decision surface* in our discussion we consider only the case of two class discrimination, in which there is a single decision surface.

When we mention a *test point*, we mean a point $x \in \mathbb{R}^n$ that does not belong to the training set $\mathcal{S}$ and for which the algorithm is to decide on a class $\tilde{f}(x)$.

By *distance*, we mean the usual Euclidean distance in input-space $\mathbb{R}^n$. The distance between two points $a$ and $b$ will be written $d(a, b)$ or alternatively $\|a - b\|$.

The distance between a single point $x$ and a set of points $S$ is the distance to the closest point of the set: $d(x, S) = \min_{p \in S} d(x, p)$.

The *K-neighborhood* $\mathcal{V}^K(x)$ of a test point $x$ is the set of the $K$ points of $\mathcal{S}$ whose distance to $x$ is smallest.

The *K-c-neighborhood* $\mathcal{V}_c^K(x)$ of a test point $x$ is the set of $K$ points of $\mathcal{S}_c$ whose distance to $x$ is smallest.

By *Nearest Neighbor* algorithm (1NN) we mean the following algorithm: the class of a test point $x$ is decided to be the same as the class of its closest neighbor in $S$.

By *K-Nearest Neighbor* algorithm (KNN) we mean the following algorithm: the class of a test point $x$ is decided to be the same as the class appearing most frequently among the K-neighborhood of $x$.

## 2.2 The intuition

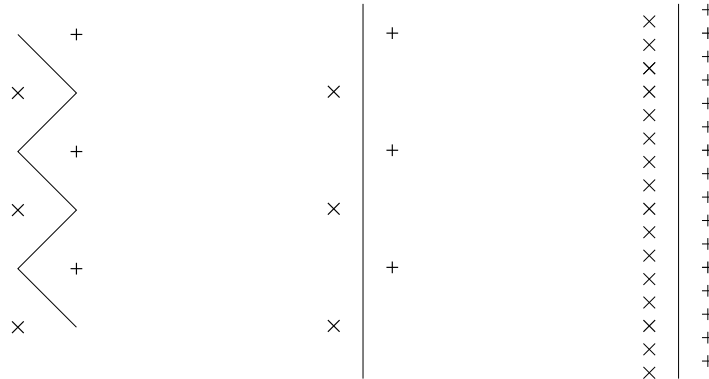

Figure 1: *A local view of the decision surface produced by the Nearest Neighbor (left) and SVM (center) algorithms, and how the Nearest Neighbor solution gets closer to the SVM solution in the limit, if the support for the density of each class is a manifold which can be considered locally linear (right).*

Figure 1 illustrates a possible intuition about why SVMs outperforms 1NNs when we have a finite number of samples. For classification tasks where the classes are considered to be mostly separable,[2] we often like to think of each class as residing close to a lower-dimensional manifold (in the high dimensional input space) which can reasonably be considered locally linear[3]. In the case of a finite number of samples, "missing" samples would appear as "holes" introducing artifacts in the decision surface produced by classical Nearest Neighbor algorithms. Thus the decision surface, while having the largest possible *local margin* with regard to the training points, is likely to have a poor small *local margin* with respect to yet unseen samples falling close to the locally linear manifold, and will thus result in poor generalization performance. This problem fundamentally remains with the K Nearest Neighbor (KNN) variant of the algorithm, but, as can be seen on the figure, it does not seem to affect the decision surface produced by SVMs (as the surface is constrained to a particular smooth form, a straight line or hyperplane in the case of linear SVMs). It is interesting to notice, if the assumption of locally linear class manifolds holds, how the 1NN solution approaches the SVM solution in the limit as we increase the number of samples.

To fix this problem, the idea is to somehow *fantasize* the missing points, based on a local linear approximation of the manifold of each class. This leads to modified Nearest Neighbor algorithms described in the next sections.[4]

### 2.3 The basic algorithm

Given a test point $x$, we are really interested in finding the closest neighbor, not among the training set $\mathcal{S}$, but among an abstract, virtually enriched training set that would contain all the *fantasized* "missing" points of the manifold of each class, locally approximated by an affine subspace. We shall thus consider, for each class $c$, the local affine subspace that passes through the $K$ points of the K-c neighborhood of $x$. This affine subspace is typically $K - 1$ dimensional or less, and we will somewhat abusively call it the "local hyperplane".[5]

Formally, the local hyperplane can be defined as

$$LH_c^K(x) = \left\{ p \mid p = \sum_{k=1}^{K} \alpha_k N_k, \ \alpha \in \mathbb{R}^K, \ \sum_{k=1}^{K} \alpha_k = 1 \right\} \tag{1}$$

where $\{N_1, \ldots, N_k\} = \mathcal{V}_c^K(x)$.

Another way to define this hyperplane, that gets rid of the constraint $\sum \alpha_k = 1$, is to take a reference point within the hyperplane as an origin, for instance the centroid[6] $\bar{N} = \frac{1}{K}\sum_{k=1}^{K} N_k$. This same hyperplane can then be expressed as

$$LH_c^K(x) = \left\{ p \mid p = \bar{N} + \sum_{k=1}^{K} \alpha_k \overrightarrow{V}_k, \ \alpha \in \mathbb{R}^K \right\} \tag{2}$$

where $\overrightarrow{V}_k = N_k - \bar{N}$.

Our *modified nearest neighbor algorithm* then associates a test point $x$ to the class $c$ whose hyperplane $LH_c^K(x)$ is closest to $x$. Formally $\tilde{f}(x) = \arg\min_{c \in \mathcal{C}} d(x, LH_c^K(x))$, where $d(x, LH_c^K(x))$ is logically called *K-local Hyperplane Distance*, hence the name *K-local Hyperplane Distance Nearest Neighbor* algorithm (HKNN in short).

Computing, for each class $c$

$$
\begin{aligned}
d(x, LH_c^K(x)) &= \min_{p \in LH_c^K(x)} \|x - p\| \\
&= \min_{\alpha \in \mathbb{R}^K} \left\| x - \bar{N} - \sum_{k=1}^{K} \alpha_k \overrightarrow{V}_k \right\|
\end{aligned} \tag{3}
$$

amounts to solving a linear system in $\alpha$, that can be easily expressed in matrix form as:

$$(V' \cdot V) \cdot \alpha = V' \cdot (x - \bar{N}) \tag{4}$$

where $x$ and $\bar{N}$ are $n$ dimensional column vectors, $\alpha = (\alpha_1, \ldots, \alpha_K)'$, and $V$ is a $n \times K$ matrix whose columns are the $\overrightarrow{V}_k$ vectors defined earlier.[7]

## 2.4 Links with other paradigms

The proposed HKNN algorithm is very similar in spirit to the *Tangent Distance Algorithm* [13]. $LH_c^K(x)$ can be seen as a tangent hyperplane representing a set of local directions of transformation (any linear combination of the $\overrightarrow{V}_k$ vectors) that do not affect the class identity. These are *invariances*. The main difference is that in HKNN these invariances are inferred directly from the local neighborhood in the training set, whereas in Tangent Distance, they are based on prior knowledge. It should be interesting (and relatively easy) to combine both approaches for improved performance when prior knowledge is available.

Previous work on nearest-neighbor variations based on other locally-defined metrics can be found in [12, 9, 6, 7], and is very much related to the more general paradigm of *Local Learning Algorithms* [3, 1, 10].

We should also mention close similarities between our approach and the recently proposed *Local Linear Embedding* [11] method for dimensionality reduction.

The idea of fantasizing points around the training points in order to define the decision surface is also very close to methods based on estimating the class-conditional input density [14, 4].

Besides, it is interesting to look at HKNN from a different, less geometrical angle: it can be understood as choosing the class that achieves the best reconstruction (the smallest reconstruction error) of the test pattern through a linear combination of $K$ particular prototypes of that class (the $K$ neighbors). From this point of view, the algorithm is very similar to the *Nearest Feature Line* (NFL) [8] method. They differ in the fact that NFL considers all pairs of points for its search rather than the local $K$ neighbors, thus looking at many ($l^2$) lines (i.e. 2 dimensional affine subspaces), rather than at a single $K - 1$ dimensional one.

## 3 Fixing the basic HKNN algorithm

### 3.1 Problem arising for large K

One problem with the basic HKNN algorithm, as previously described, arises as we increase the value of $K$, i.e. the number of points considered in the neighborhood of the test point. In a typical high dimensional setting, *exact* colinearities between input patterns are rare, which means that as soon as $K > n$, any pattern of $\mathbb{R}^n$ (including nonsensical ones) can be produced by a linear combination of the $K$ neighbors. The "actual" dimensionality of the manifold may be much less than $K$. This is due to "near-colinearities" producing directions associated to small eigenvalues of the covariance matrix $V' \cdot V$ that are but noise, that can lead the algorithm to mistake those noise directions for "invariances", and may hurt its performance even for smaller values of $K$. Another related issue is that the linear approximation of the class manifold by a hyperplane is valid only locally, so we might want to restrict the "fantasizing" of class members to a smaller region of the hyperplane. We considered two ways of dealing with these problems.[8]

### 3.2 The convex hull solution

One way to avoid the above mentioned problems is to restrict ourselves to considering the *convex hull* of the neighbors, rather than the whole hyperplane they support (of which the convex hull is a subset). This corresponds to adding a constraint of $\alpha_k \geq 0, \forall k$ to equation (1). Unlike the problem of computing the distance to the hyperplane, the distance to the convex hull cannot be found by solving a simple linear system, but typically requires solving a quadratic programming problem (very similar to the one of SVMs). While this

is more complex to implement, it should be mentioned that the problems to be solved are of a relatively small dimension of order $K$, and that the time of the whole algorithm will very likely still be dominated by the search of the $K$ nearest neighbors within each class. This algorithm will be referred to as *K-local Convex Distance Nearest Neighbor Algorithm* (CKNN in short).

### 3.3 The "weight decay" penalty solution

This consists in incorporating a penalty term to equation (3) to penalize large values of $\alpha$ (i.e. it penalizes moving away from the centroid, especially in non essential directions):

$$d'(x, LH_c^K(x))^2 = \min_{\alpha \in \mathbb{R}^K} \left\| x - \bar{N} - \sum_{k=1}^{K} \alpha_k \overrightarrow{V}_k \right\|^2 + \lambda \sum_{k=1}^{K} \alpha_k^2 \tag{5}$$

The solution for $\alpha$ is given by solving the linear system $(V' \cdot V + \lambda I_n) \cdot \alpha = V' \cdot (x - \bar{N})$ where $I_n$ is the $n \times n$ identity matrix. This is equation (4) with an additional diagonal term. The resulting algorithm is a generalization of HKNN (basic HKNN corresponds to $\lambda = 0$).

## 4 Experimental results

We performed a number of experiments, to highlight different properties of the algorithms:
• A first 2D toy example (see Figure 2) graphically illustrates the qualitative differences in the decision surfaces produced by KNN, linear SVM and CKNN.
• Table 1 gives quantitative results on two real-world digit OCR tasks, allowing to compare the performance of the different old and new algorithms.
• Figure 3 illustrates the problem arising with large $K$, mentioned in Section 3, and shows that the two proposed solutions: CKNN and HKNN with an added weight decay $\lambda$, allow to overcome it.
• In our final experiment, we wanted to see if the good performance of the new algorithms absolutely depended on having all the training points at hand, as this has a direct impact on speed. So we checked what performance we could get out of HKNN and CKNN when using only a small but representative subset of the training points, namely the set of support vectors found by a Gaussian Kernel SVM. The results obtained for MNIST are given in Table 2, and look very encouraging. HKNN appears to be able to perform as well or better than SVMs *without* requiring more data points than SVMs.

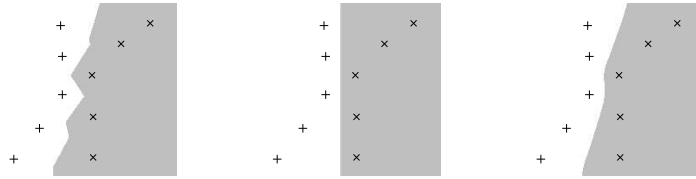

Figure 2: *2D illustration of the decision surfaces produced by KNN (left, K=1), linear SVM (middle), and CKNN (right, K=2). The "holes" are again visible in KNN. CKNN doesn't suffer from this, but keeps the objective of* **maximizing the margin locally**.

## 5 Conclusion

From a few geometrical intuitions, we have derived two modified versions of the KNN algorithm that look very promising. HKNN is especially attractive: it is very simple to implement on top of a KNN system, as it only requires the additional step of solving a small and simple linear system, and appears to greatly boost the performance of standard KNN even above the level of SVMs. The proposed algorithms share the advantages of KNN (no training required, ideal for fast adaptation, natural handling of the multi-class case) and its drawbacks (requires large memory, slow testing). However our latest result also indicate the possibility of substantially reducing the reference set in memory without loosing on accuracy. This suggests that the algorithm indeed captures essential information in the data, and that our initial intuition on the nature of the flaw of KNN may well be at least partially correct.

## Footnotes

[1]for the purpose of this discussion, we consider the original hard-margin SVM algorithm for two linearly separable classes.

[2]By 'mostly separable" we mean that the Bayes error is almost zero, and the optimal decision surface has not too many disconnected components.

[3]i.e. each class has a probability density with a 'support" that is a lower-dimensional manifold, and with the probability quickly fading, away from this support.

[4]Note that although we never generate the 'fantasy" points explicitly, the proposed algorithms are really equivalent to classical 1NN with fantasized points.

[5]Strictly speaking a hyperplane in an $n$ dimensional input space is an $n-1$ affine subspace, while our "local hyperplanes" can have fewer dimensions.

[6]We could be using one of the $K$ neighbors as the reference point, but this formulation with the centroid will prove useful later.

[7]Actually there is an infinite number of solutions to this system since the $\overrightarrow{V}_k$ are linearly dependent: remember that the initial formulation had an equality constraint and thus only $K - 1$ effective degrees of freedom. But we are interested in $d(x, LH_c^K(x))$ not in $\alpha$ so any solution will do. Alternatively, we can remove one of the $\overrightarrow{V}_k$ from the system so that it has a unique solution.

[8]A third interesting avenue, which we did not have time to explore, would be to keep only the most relevant principal components of $V$, ignoring those corresponding to small eigenvalues.

## References

[1] C. G. Atkeson, A. W. Moore, and S. Schaal. Locally weighted learning. *Artificial Intelligence Review*, 1996.

[2] B. Boser, I. Guyon, and V. Vapnik. An algorithm for optimal margin classifiers. In *Fifth Annual Workshop on Computational Learning Theory*, pages 144–152, Pittsburgh, 1992.

[3] L. Bottou and V. Vapnik. Local learning algorithms. *Neural Computation*, 4(6):888–900, 1992.

[4] Olivier Chapelle, Jason Weston, L´eon Bottou, and Vladimir Vapnik. Vicinal risk minimization. In T.K. Leen, T.G. Dietterich, and V. Tresp, editors, *Advances in Neural Information Processing Systems*, volume 13, pages 416–422, 2001.

[5] T.M. Cover and P.E. Hart. Nearest neighbor pattern classification. *IEEE Transactions on Information Theory*, 13(1):21–27, 1967.

[6] J. Friedman. Flexible metric nearest neighbor classification. Technical Report 113, Stanford University Statistics Department, 1994.

[7] Trevor Hastie and Robert Tibshirani. Discriminant adaptive nearest neighbor classification and regression. In David S. Touretzky, Michael C. Mozer, and Michael E. Hasselmo, editors, *Advances in Neural Information Processing Systems*, volume 8, pages 409–415. The MIT Press, 1996.

[8] S.Z. Li and J.W. Lu. Face recognition using the nearest feature line method. *IEEE Transactions on Neural Networks*, 10(2):439–443, 1999.

[9] J. Myles and D. Hand. The multi-class measure problem in nearest neighbour discrimination rules. *Pattern Recognition*, 23:1291–1297, 1990.

[10] D. Ormoneit and T. Hastie. Optimal kernel shapes for local linear regression. In S. A. Solla, T. K. Leen, and K-R. Mller, editors, *Advances in Neural Information Processing Systems*, volume 12. MIT Press, 2000.

[11] Sam Roweis and Lawrence Saul. Nonlinear dimensionality reduction by locally linear embedding. *Science*, 290(5500):2323–2326, Dec. 2000.

[12] R. D. Short and K. Fukunaga. The optimal distance measure for nearest neighbor classification. *IEEE Transactions on Information Theory*, 27:622–627, 1981.

[13] P. Y. Simard, Y. A. LeCun, J. S. Denker, and B. Victorri. Transformation invariance in pattern recognition — tangent distance and tangent propagation. *Lecture Notes in Computer Science*, 1524, 1998.

[14] S. Tong and D. Koller. Restricted bayes optimal classifiers. In *Proceedings of the 17th National Conference on Artificial Intelligence (AAAI)*, pages 658–664, Austin, Texas, 2000.

[15] V.N. Vapnik. *The Nature of Statistical Learning Theory*. Springer, New York, 1995.

[16] Bin Zhang. Is the maximal margin hyperplane special in a feature space? Technical Report HPL-2001-89, Hewlett-Packards Labs, 2001.

Table 1: *Test-error obtained on the USPS and MNIST digit classification tasks by KNN, SVM (using a Gaussian Kernel), HKNN and CKNN. Hyper parameters were tuned on a separate validation set. Both HKNN and CKNN appear to perform much better than original KNN, and even compare favorably to SVMs.*

| Data Set | Algorithm | Test Error | Parameters used |
|---|---|---|---|
| USPS | KNN | 4.98% | $K = 1$ |
| (6291 train, | SVM | 4.33% | $\sigma = 8, C = 100$ |
| 1000 valid., | HKNN | 3.93% | $K = 15, \lambda = 30$ |
| 2007 test points) | CKNN | 3.98% | $K = 20$ |
| MNIST | KNN | 2.95% | $K = 3$ |
| (50000 train, | SVM | 1.30% | $\sigma = 6.47, C = 100$ |
| 10000 valid., | HKNN | 1.26% | $K = 65, \lambda = 10$ |
| 10000 test points) | CKNN | 1.46% | $K = 70$ |

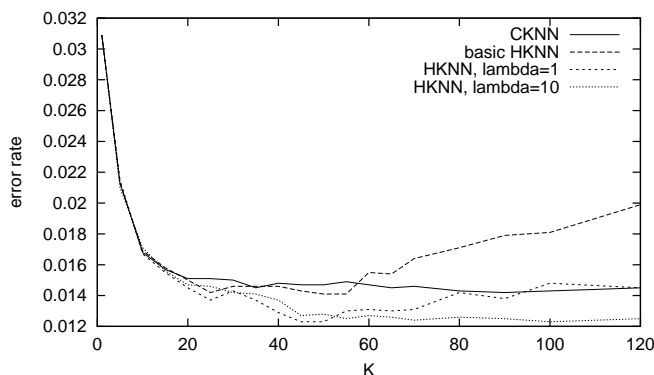

Figure 3: *Error rate on MNIST as a function of $K$ for CKNN, and HKNN with different values of $\lambda$. As can be seen the basic HKNN algorithm performs poorly for large values of $K$. As expected, CKNN is relatively unaffected by this problem, and HKNN can be made robust through the added 'weight decay" penalty controlled by $\lambda$.*

Table 2: *Test-error obtained on MNIST with HKNN and CKNN when using a reduced training set made of the 16712 support vectors retained by the best Gaussian Kernel SVM. This corresponds to 28% of the initial 60000 training patterns. Performance is even better than when using the whole dataset. But here, hyper parameters $K$ and $\lambda$ were chosen with the test set, as we did not have a separate validation set in this setting. It is nevertheless remarkable that comparable performances can be achieved with far fewer points.*

| Data Set | Algorithm | Test Error | Parameters used |
|---|---|---|---|
| MNIST (16712 train s.v., | HKNN | 1.23% | $K = 60, \lambda = 10$ |
| 10000 test points) | CKNN | 1.36% | $K = 45$ |